# Unsupervised On-Line Learning of Decision Trees for Hierarchical Data Analysis

**Marcus Held and Joachim M. Buhmann**
Rheinische Friedrich–Wilhelms–Universität
Institut für Informatik III, Römerstraße 164
D-53117 Bonn, Germany
email: {held, jb}.cs.uni-bonn.de
WWW: http://www-dbv.cs.uni-bonn.de

## Abstract

An adaptive on–line algorithm is proposed to estimate hierarchical data structures for non–stationary data sources. The approach is based on the principle of *minimum cross entropy* to derive a decision tree for data clustering and it employs a *metalearning* idea (learning to learn) to adapt to changes in data characteristics. Its efficiency is demonstrated by grouping non–stationary artifical data and by hierarchical segmentation of LANDSAT images.

## 1  Introduction

Unsupervised learning addresses the problem to detect structure inherent in unlabeled and unclassified data. The simplest, but not necessarily the best approach for extracting a grouping structure is to represent a set of data samples $\mathcal{X} = \left\{ x_i \in \mathbb{R}^d | i = 1, \dots, N \right\}$ by a set of prototypes $\mathcal{Y} = \left\{ y_\alpha \in \mathbb{R}^d | \alpha = 1, \dots, K \right\}$, $K \ll N$. The encoding usually is represented by an assignment matrix $\mathbf{M} = (M_{i\alpha})$, where $M_{i\alpha} = 1$ if and only if $x_i$ belongs to cluster $\alpha$, and $M_{i\alpha} = 0$ otherwise. According to this encoding scheme, the cost function $\mathcal{H}(\mathbf{M}, \mathcal{Y}) = \frac{1}{N} \sum_{i=1}^{N} M_{i\alpha} \mathcal{D}(x_i, y_\alpha)$ measures the quality of a data partition, i.e., optimal assignments and prototypes $(\mathbf{M}, \mathcal{Y})^{\text{opt}} = \arg\min_{\mathbf{M}, \mathcal{Y}} \mathcal{H}(\mathbf{M}, \mathcal{Y})$ minimize the inhomogeneity of clusters w.r.t. a given distance measure $\mathcal{D}$. For reasons of simplicity we restrict the presentation to the *sum–of–squared–error criterion* $\mathcal{D}(x, y) = \|x - y\|^2$ in this paper. To facilitate this minimization a deterministic annealing approach was proposed in [5] which maps the discrete optimization problem, i.e. how to determine the data assignments, via the *Maximum Entropy Principle* [2] to a continuous parameter es-

timation problem. Deterministic annealing introduces a Lagrange multiplier $\beta$ to control the approximation of $\mathcal{H}(\mathbf{M}, \mathcal{Y})$ in a probabilistic sense. Equivalently to maximize the entropy at fixed expected $K$-means costs we minimize the free energy $\mathcal{F} = \frac{1}{\beta} \sum_{i=1}^{N} \ln \left( \sum_{\mu=1}^{K} \exp\left(-\beta\mathcal{D}\left(x_i, y_\alpha\right)\right) \right)$ w.r.t. the prototypes $y_\alpha$. The assignments $M_{i\alpha}$ are treated as random variables yielding a fuzzy centroid rule

$$y_\alpha = \sum_{i=1}^{N} \langle M_{i\alpha}\rangle x_i / \sum_{i=1}^{N} \langle M_{i\alpha}\rangle, \tag{1}$$

where the expected assignments $\langle M_{i\alpha}\rangle$ are given by Gibbs distributions

$$\langle M_{i\alpha}\rangle = \frac{\exp\left(-\beta\mathcal{D}\left(x_i, y_\alpha\right)\right)}{\sum_{\mu=1}^{K} \exp\left(-\beta\mathcal{D}\left(x_i, y_\alpha\right)\right)}. \tag{2}$$

For a more detailed discussion of the DA approach to data clustering cf. [1, 3, 5].

In addition to assigning data to clusters (1,2), hierarchical clustering provides the partitioning of data space with a tree structure. Each data sample $x$ is sequentially assigned to a nested structure of partitions which hierarchically cover the data space $\mathbb{R}^d$. This sequence of special decisions is encoded by decision rules which are attached to nodes along a path in the tree (see also fig. 1).

Therefore, learning a decision tree requires to determine a tree topology, the accompanying assignments, the inner node labels $S$ and the prototypes $\mathcal{Y}$ at the leaves. The search of such a hierarchical partition of the data space should be guided by an optimization criterion, i.e., minimal distortion costs.

This problem is solvable by a two–stage approach, which on the one hand minimizes the distortion costs at the leaves given the tree structure and on the other hand optimizes the tree structure given the leaf induced partition of $\mathbb{R}^d$. This approach, due to Miller & Rose [3], is summarized in section 2. The extensions for adaptive on-line learning and experimental results are described in sections 3 and 4, respectively.

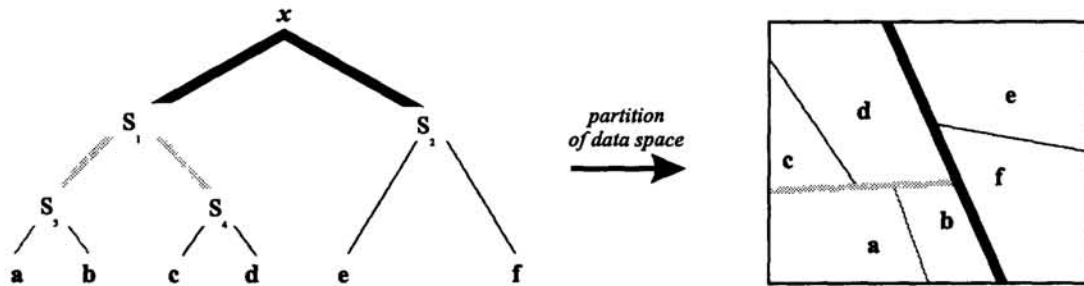

Figure 1: Right: Topology of a decision tree. Left: Induced partitioning of the data space (positions of the letters also indicate the positions of the prototypes). Decisions are made according to the nearest neighbor rule.

## 2   Unsupervised Learning of Decision Trees

Deterministic annealing of hierarchical clustering treats the assignments of data to inner nodes of the tree in a probabilistic way analogous to the expected assignments of data to leaf prototypes. Based on the maximum entropy principle, the probability $\Phi_{i,j}^{\mathrm{H}}$ that data point $x_i$ reaches inner node $s_j$ is recursively defined by (see [3]):

$$\Phi_{i,\mathrm{root}}^{\mathrm{H}} := 1, \quad \Phi_{i,j}^{\mathrm{H}} = \Phi_{i,\mathrm{parent}(j)}^{\mathrm{H}}\pi_{i,j}, \quad \pi_{i,j} = \frac{\exp\left(-\gamma\mathcal{D}\left(x_i, s_j\right)\right)}{\sum\limits_{k\in\mathrm{siblings}(j)} \exp\left(-\gamma\mathcal{D}\left(x_i, s_k\right)\right)}, \tag{3}$$

where the Lagrange multiplier $\gamma$ controls the fuzziness of all the transitions $\pi_{i,j}$. On the other hand, given the tree topology and the prototypes at the leaves, the maximum entropy principle naturally recommends an *ideal probability* $\Phi_{i,\alpha}^{\mathrm{I}}$ at leaf $y_\alpha$, resp. at an inner node $s_j$,

$$\Phi_{i,\alpha}^{\mathrm{I}} = \frac{\exp\left(-\beta\mathcal{D}\left(x_i, y_\alpha\right)\right)}{\sum\limits_{\mu\in\mathcal{Y}} \exp\left(-\beta\mathcal{D}\left(x_i, y_\mu\right)\right)} \quad \text{and} \quad \Phi_{i,j}^{\mathrm{I}} = \sum_{k\in\mathrm{descendants}(j)} \Phi_{i,k}^{\mathrm{I}}. \tag{4}$$

We apply the principle of minimum cross entropy for the calculation of the prototypes at the leaves given a priori the probabilities for the parents of the leaves. Minimization of the cross entropy with fixed expected costs $\langle H_{x_i}\rangle = \sum_\alpha \langle M_{i\alpha}\rangle \mathcal{D}\left(x_i, y_\alpha\right)$ for the data point $x_i$ yields the expression

$$\min_{\{\langle M_{i\alpha}\rangle\}} \mathcal{I}\left(\{\langle M_{i\alpha}\rangle\}\|\{\Phi_{i,\mathrm{parent}(\alpha)}^{\mathrm{H}}/K\}\right) = \min_{\{\langle M_{i\alpha}\rangle\}} \sum_\alpha \langle M_{i\alpha}\rangle \ln \frac{\langle M_{i\alpha}\rangle}{\Phi_{i,\mathrm{parent}(\alpha)}^{\mathrm{H}}}, \tag{5}$$

where $\mathcal{I}$ denotes the Kullback–Leibler divergence and $K$ defines the degree of the inner nodes. The *tilted distribution*

$$\langle M_{i\alpha}\rangle = \frac{\Phi_{i,\mathrm{parent}(\alpha)}^{\mathrm{H}} \exp\left(-\beta\mathcal{D}\left(x_i, y_\alpha\right)\right)}{\sum_\mu \Phi_{i,\mathrm{parent}(\mu)}^{\mathrm{H}} \exp\left(-\beta\mathcal{D}\left(x_i, y_\mu\right)\right)}. \tag{6}$$

generalizes the probabilistic assignments (2). In the case of Euclidian distances we again obtain the centroid formula (1) as the minimum of the free energy $\mathcal{F} = -\frac{1}{\beta} \sum_{i=1}^{N} \ln\left[\sum_{\alpha\in\mathcal{Y}} \Phi_{i,\mathrm{parent}(\alpha)}^{\mathrm{H}} \exp\left(-\beta\mathcal{D}\left(x_i, y_\alpha\right)\right)\right]$. Constraints induced by the tree structure are incorporated in the assignments (6). For the optimization of the hierarchy, Miller and Rose in a second step propose the minimization of the distance between the hierarchical probabilities $\Phi_{\cdot,\cdot}^{\mathrm{H}}$ and the ideal probabilities $\Phi_{\cdot,\cdot}^{\mathrm{I}}$, the distance being measured by the Kullback–Leibler divergence

$$\min_{\gamma,\mathcal{S}} \sum_{s_j\in\mathrm{parent}(\mathcal{Y})} \mathcal{I}\left(\{\Phi_{\cdot,j}^{\mathrm{I}}\}\|\{\Phi_{\cdot,j}^{\mathrm{H}}\}\right) \equiv \min_{\gamma,\mathcal{S}} \sum_{s_j\in\mathrm{parent}(\mathcal{Y})} \sum_{i=1}^{N} \Phi_{i,j}^{\mathrm{I}} \ln \frac{\Phi_{i,j}^{\mathrm{I}}}{\Phi_{i,j}^{\mathrm{H}}}. \tag{7}$$

Equation (7) describes the minimization of the sum of cross entropies between the probability densities $\Phi_{\cdot,\cdot}^{\mathrm{I}}$ and $\Phi_{\cdot,\cdot}^{\mathrm{H}}$ over the parents of the leaves. Calculating the gradients for the inner nodes $s_j$ and the Lagrange multiplier $\gamma$ we receive

$$\frac{\partial}{\partial s_j}\mathcal{I} = -2\gamma \sum_{i=1}^{N} (x_i - s_j)\left\{\Phi_{i,j}^{\mathrm{I}} - \Phi_{i,\mathrm{parent}(j)}^{\mathrm{I}}\pi_{i,j}\right\} := -2\gamma \sum_{i=1}^{N} \Delta_1\left(x_i, s_j\right), \tag{8}$$

$$\frac{\partial}{\partial\gamma}\mathcal{I} = \sum_{i=1}^{N}\sum_{j\in\mathcal{S}} \mathcal{D}\left(x_i, s_j\right)\left\{\Phi_{i,j}^{\mathrm{I}} - \Phi_{i,\mathrm{parent}(j)}^{\mathrm{I}}\pi_{i,j}\right\} := \sum_{i=1}^{N}\sum_{j\in\mathcal{S}} \Delta_2\left(x_i, s_j\right). \tag{9}$$

The first gradient is a weighted average of the difference vectors $(x_i - s_j)$, where the weights measure the mismatch between the probability $\Phi_{i,j}^{\mathrm{I}}$ and the probability induced by the transition $\pi_{i,j}$. The second gradient (9) measures the scale $-\mathcal{D}\left(x_i, s_j\right)$ – on which the transition probabilities are defined, and weights them with the mismatch between the ideal probabilities. This procedure yields an algorithm which starts at a small value $\beta$ with a complete tree and identical test vectors attached to all nodes. The prototypes at the leaves are optimized according to (6) and the centroid rule (1), and the hierarchy is optimized by (8) and (9). After convergence one increases $\beta$ and optimizes the hierarchy and the prototypes at the leaves again. The increment of $\beta$ leads to phase transitions where test vectors separate from each other and the formerly completely degenerated tree evolves its structure. For a detailed description of this algorithm see [3].

# 3 On-Line Learning of Decision Trees

Learning of decision trees is refined in this paper to deal with unbalanced trees and on-line learning of trees. Updating identical nodes according to the gradients (9) with assignments (6) weighs parameters of unbalanced tree structures in an unsatisfactory way. A detailed analysis reveals that degenerated test vectors, i.e., test vectors with identical components, still contribute to the assignments and to the evolution of $\gamma$. This artefact is overcome by using dynamic tree topologies instead of a predefined topology with indistinguishable test vectors. On the other hand, the development of an on–line algorithm makes it possible to process huge data sets and non–stationary data. For this setting there exists the need of on–line learning rules for the prototypes at the leaves, the test vectors at the inner nodes and the parameters $\gamma$ and $\beta$. Unbalanced trees also require rules for splitting and merging nodes.

Following Buhmann and Kühnel [1] we use an expansion of order $O(1/n)$ of (1) to estimate the prototypes for the $N$th datapoint

$$y_\alpha^N \approx y_\alpha^{N-1} + \eta_\alpha \frac{\langle M_{N\alpha}^{N-1} \rangle}{p_\alpha^{N-1} M} \left( x_N - y_\alpha^{N-1} \right), \tag{10}$$

where $p_\alpha^N \approx p_\alpha^{N-1} + 1/M \left( \langle M_{N\alpha}^{N-1} \rangle - p_\alpha^{N-1} \right)$ denotes the probability of the occurence of class $\alpha$. The parameters $M$ and $\eta_\alpha$ are introduced in order to take the possible non–stationarity of the data source into account. $M$ denotes the size of the data window, and $\eta_\alpha$ is a node specific learning rate.

Adaptation of the inner nodes and of the parameter $\gamma$ is performed by stochastic approximation using the gradients (8) and (9)

$$s_j^N \quad := \quad s_j^{N-1} + \eta_j \gamma^{N-1} \Delta_1 \left( x_N, s_j^{N-1} \right), \tag{11}$$

$$\gamma^N \quad := \quad \gamma^{N-1} - \eta_\gamma \sum_{s_j \in \mathcal{S}} \Delta_2 \left( x_N, s_j^{N-1} \right). \tag{12}$$

For an appropriate choice of the learning rates $\eta$, the *learning to learn* approach of Murata et al. [4] suggests the learning algorithm

$$w^N = w^{N-1} - \eta^{N-1} f \left( x_N, w^{N-1} \right). \tag{13}$$

The flow $f$ in parameter space determines the change of $w^{N-1}$ given a new datapoint $x_N$. Murata et al. derive the following update scheme for the learning rate:

$$r^N \quad = \quad (1 - \delta) r^{N-1} + \delta f \left( x_N, w^{N-1} \right), \tag{14}$$

$$\eta^N \quad = \quad \eta^{N-1} + \nu_1 \eta^{N-1} \left( \nu_2 \| r^N \| - \eta^{N-1} \right), \tag{15}$$

where $\nu_1, \nu_2$ and $\delta$ are control parameters to balance the tradeoff between accuracy and convergence rate. $r^N$ denotes the leaky average of the flow at time $N$.

The adaptation of $\beta$ has to observe the necessary condition for a phase transition $\beta > \beta_{\text{crit}} \equiv 1/2\delta_{\text{max}}$, $\delta_{\text{max}}$ being the largest eigenvalue of the covariance matrix [3]

$$\Sigma_\alpha = \sum_{i=1}^{M} (x_i - y_\alpha) (x_i - y_\alpha)^t \langle M_{i\alpha} \rangle / \sum_{i=1}^{M} \langle M_{i\alpha} \rangle. \tag{16}$$

Rules for splitting and merging nodes of the tree are introduced to deal with unbalanced trees and non–stationary data. Simple rules measure the distortion costs at the prototypes of the leaves. According to these costs the leaf with highest

distortion costs is split. The merging criterion combines neighboring leaves with minimal distance in a greedy fashion. The parameter $M$ (10), the typical time scale for changes in the data distribution is used to fix the time between splitting resp. merging nodes and the update of $\beta$. Therefore, $M$ controls the time scale for changes of the tree topology. The learning parameters for the learning to learn rules (13)-(15) are chosen empirically and are kept constant for all experiments.

## 4   Experiments

The first experiment demonstrates how a drifting two dimensional data source can be tracked. This data source is generated by a fixed tree augmented with transition probabilities at the edges and with Gaussians at the leaves. By descending the tree structure this generates an i.i.d. random variable $X \in \mathbb{R}^2$, which is rotated around the origin of $\mathbb{R}^2$ to obtain a random variable $T(N) = R(\omega, N)X$. $R$ is an orthogonal matrix, $N$ denotes the number of the actual data point and $\omega$ denotes the angular velocity, $M = 500$. Figure 2 shows 45 degree snapshots of the learning of this non-stationary data source. We start to take these snapshots after the algorithm has developed its final tree topology (after $\approx 8000$ datapoints). Apart from fluctuations of the test vectors at the leaves, the whole tree structure is stable while tracking the rotating data source.

Additional experiments with higher dimensional data sources confirm the robustness of the algorithm w.r.t. the dimension of the data space, i. e. similiar tracking performances for different dimensions are observed, where differences are explained as differences in the data sources (figure 3). This performance is measured by the variance of the mean of the distances between the data source trajectory and the trajectories of the test vectors at the nodes of the tree.

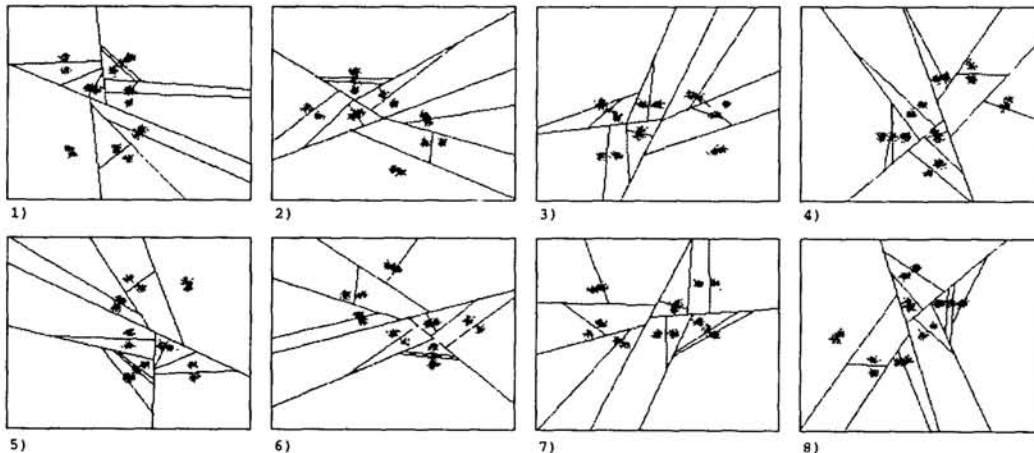

Figure 2: 45 degree snapshots of the learning of a data source which rotates with a velocity $\omega = 2\pi/30000$ (360 degree per 30000 data samples).

A second experiment demonstrates the learning of a switching data source. The results confirm a good performance concerning the restructuring of the tree (see figure 4). In this experiment the algorithm learns a given data source and after 10000 data points we switch to a different source.

As a real-world example of on-line learning of huge data sources the algorithm is applied to the hierarchical clustering of 6-dimensional LANDSAT data. The heat

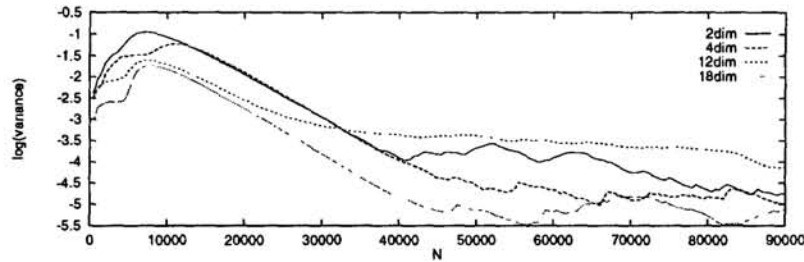

Figure 3: Tracking performance for different dimensions. As data sources we use $d$–dimensional Gaussians which are attached to a unit sphere. To the components of every random sample $X$ we add $\sin(\omega N)$ in order to introduce non stationarity. The first 8000 samples are used for the development of the tree topology.

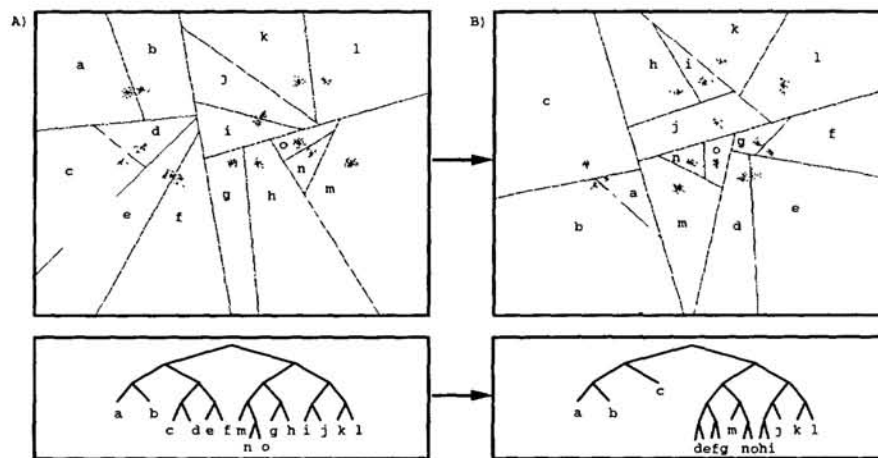

Figure 4: Learning a switching data source: top: a) the partition of the data space after 10000 data samples given the first source, b) the restructured partition after additional 2500 samples. Below: accompanying tree topologies.

channel has been discarded because of its reduced resolution. In a preprocessing step all channels are rescaled to unit variance, which alternatively could be established by using a Mahalanobis distance. Note that the decision tree which clusters this data supplies us with a hierarchical segmentation of the corresponding LANDSAT image. A tree of 16 leaves has been learned on a training set of $128 \times 128$ data samples, and it has been applied to a test set of $128 \times 128$ LANDSAT pixels. The training is established by 15 sequential runs through the test set, where after each $M = 16384$ run a split of one node is carried out. The resulting empirical errors (0.49 training distortion and 0.55 test distortion) differ only slightly from the errors obtained by the LBG algorithm applied to the whole training set (0.42 training distortion and 0.52 test distortion). This difference is due to the fact that not every data point is assigned to the nearest leaf prototype by a decision tree induced partition. The segmentation of the test image is depicted in figure 5.

## 5   Conclusion

This paper presents a method for unsupervised on–line learning of decision trees. We overcome the shortcomings of the original decision tree approach and extend

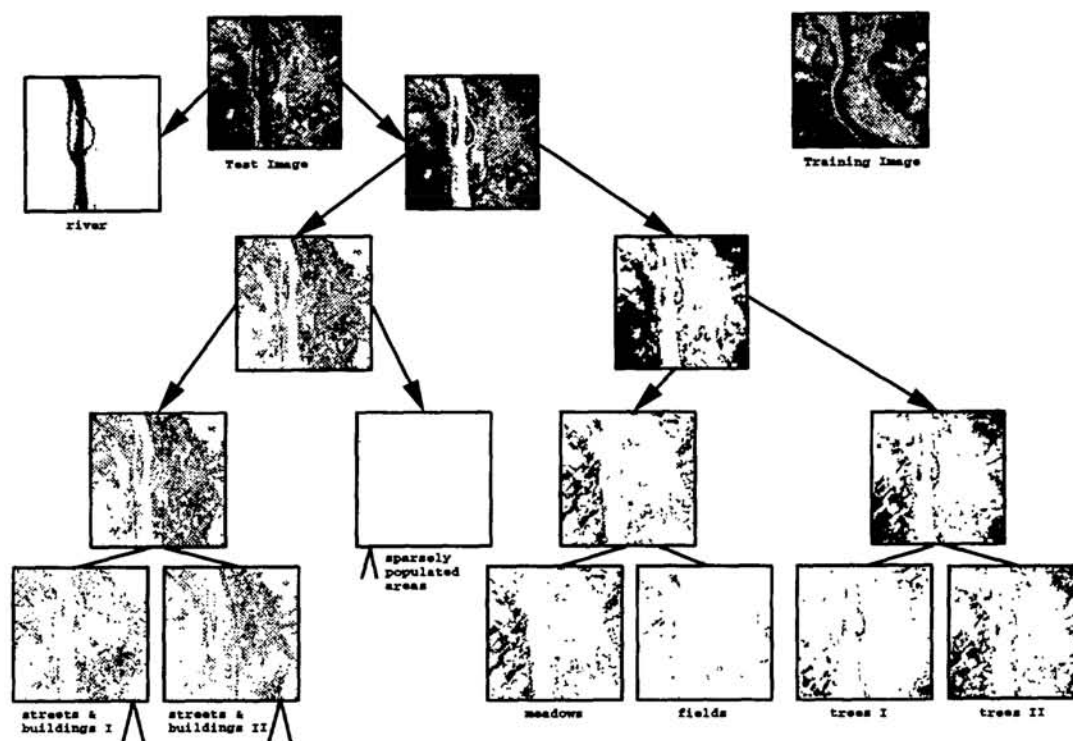

Figure 5: Hierarchical segmentation of the test image. The root represents the original image, i.e., the gray scale version of the three color channels.

it to the realm of on–line learning of huge data sets and of adaptive learning of non–stationary data. Our experiments demonstrate that the approach is capable of tracking gradually changing or switching environments. Furthermore, the method has been successfully applied to the hierarchical segmentation of LANDSAT images. Future work will address active data selection issues to significantly reduce the uncertainty of the most likely tree parameters and the learning questions related to different tree topologies.

**Acknowledgement**: This work has been supported by the German Israel Foundation for Science and Research Development (GIF) under grant #I–0403–001.06/95 and by the Federal Ministry for Education, Science and Technology (BMBF #01 M 3021 A/4).

# References

[1] J. M. Buhmann and H. Kühnel. Vector quantization with complexity costs. *IEEE Transactions on Information Theory*, 39(4):1133–1145, July 1993.

[2] T.M. Cover and J. Thomas. *Elements of Information Theory*. Wiley & Sons, 1991.

[3] D. Miller and K. Rose. Hierarchical unsupervised learning with growing via phase transitions. *Neural Computation*, 8:425–450, February 1996.

[4] N. Murata, K.-R. Müller, A. Ziehe, and S. Amari. Adaptive on-line learning in changing environments. In M.C. Mozer, M.I. Jordan, and T. Petsche, editors, *Advances in Neural Information Processing Systems*, number 9, pages 599–605. MIT Press, 1997.

[5] K. Rose, E. Gurewitz, and G.C. Fox. A deterministic annealing approach to clustering. *Pattern Recognition Letters*, 11(9):589–594, September 1990.